# Transferring Expectations in Model-based Reinforcement Learning

**Trung Thanh Nguyen, Tomi Silander, Tze-Yun Leong**
School of Computing
National University of Singapore
Singapore, 117417
{nttrung, silander, leongty}@comp.nus.edu.sg

## Abstract

We study how to automatically select and adapt multiple abstractions or representations of the world to support model-based reinforcement learning. We address the challenges of transfer learning in heterogeneous environments with varying tasks. We present an efficient, online framework that, through a sequence of tasks, learns a set of relevant representations to be used in future tasks. Without predefined mapping strategies, we introduce a general approach to support transfer learning across different state spaces. We demonstrate the potential impact of our system through improved jumpstart and faster convergence to near optimum policy in two benchmark domains.

## 1 Introduction

In reinforcement learning (RL), an agent autonomously learns how to make optimal sequential decisions by interacting with the world. The agent's learned knowledge, however, is task and environment specific. A small change in the task or the environment may render the agent's accumulated knowledge useless; costly re-learning from scratch is often needed.

Transfer learning addresses this shortcoming by accumulating knowledge in forms that can be reused in new situations. Many existing techniques assume the same state space or state representation in different tasks. While recent efforts have addressed inter-task transfer in different action or state spaces, specific mapping criteria have to be established through policy reuse [7], action correlation [14], state abstraction [22], inter-space relation [16], or other methods. Such mappings are hard to define when the agent operates in complex environments with large state spaces and multiple goal states, with possibly different state feature distributions and world dynamics. To efficiently accomplish varying tasks in heterogeneous environments, the agent has to learn to focus *attention* on the crucial features of each environment.

We propose a system that tries to transfer old knowledge, but at the same time evaluates new options to see if they work better. The agent gathers experience during its lifetime and enters a new environment equipped with *expectations* on how different aspects of the world affect the outcomes of the agent's actions. The main idea is to allow an agent to collect a library of world models or representations, called *views*, that it can consult to focus its attention in a new task. In this paper, we concentrate on approximating the transition model. The reward model library can be learned in an analogous fashion. Effective utilization of the library of world models allows the agent to capture the transition dynamics of the new environment quickly; this should lead to a jumpstart in learning and faster convergence to a near optimal policy. A main challenge is in learning to select a proper view for a new task in a new environment, without any predefined mapping strategies.

We will next formalize the problem and describe the method of collecting views into a library. We will then present an efficient implementation of the proposed transfer learning technique. After

discussing related work, we will demonstrate the efficacy of our system through a set of experiments in two different benchmark domains.

## 2 Method

In RL, a task environment is typically modeled as a Markov decision process (MDP) defined by a tuple $(S, A, T, R)$, where $S$ is a set of states; $A$ is a set of actions; $T : S \times A \times S \to [0, 1]$ is transition function, such that $T(s, a, s') = P(s'|s, a)$ indicates the probability of transiting to a state $s'$ upon taking an action $a$ in state $s$; $R : S \times A \to \mathcal{R}$ is a reward function indicating immediate expected reward after an action $a$ is taken in state $s$. The goal is then to find a policy $\pi$ that specifies an action to perform in each state so that the expected accumulated future reward (possibly giving higher weights to more immediate rewards) for each state is maximized [18]. In model-based RL, the optimal policy is calculated based on the estimates of the transition model $T$ and the reward model $R$ which are obtained by interacting with the environment.

A key idea of this work is that the agent can represent the world dynamics from its sensory state space in different ways. Such different views correspond to the agent's decisions to focus attention on only some features of the state in order to quickly approximate the state transition function.

### 2.1 Decomposition of transition model

To allow knowledge transfer from one state space to another, we assume that each state $s$ in all the state spaces can be *characterized* by a $d$-dimensional feature vector $f(s) \in \mathbb{R}^d$. The states themselves may or may not be factored. We use the idea in situation calculus [11] to decompose the transition model $T$ in accordance with the possible action effects. In the RL context, an action will stochastically create an effect that determines how the current state changes to the next one [2, 10, 14]. For example, an attempt to move left in a grid world may cause the agent to move one step left or one step forward, with small probabilities. The relative changes in states, "moved left" and "moved forward", are called *effects* of the action.

Formally, let us call MDP with a decomposed transition model CMDP (situation Calculus MDP). CMDP is defined by a tuple $(S, A, E, \tau, \eta, f, R)$ in which the transition model $T$ has been replaced by the the terms $E, \tau, \eta, f$, where $E$ is an effect set and $f$ is a function from states to their feature vectors. $\tau : S \times A \times E \to [0, 1]$ is an action model such that $\tau(s, a, e) = P(e \mid f(s), a)$ indicates the probability of achieving effect $e$ upon performing action $a$ at state $s$. Notice that the probability of effect $e$ depends on state $s$ only through the features $f(s)$. While the agent needs to learn the effects of the action, it is usually assumed to understand the meaning of the effects, i.e., how the effects turn each state into a next state. This knowledge is captured in a deterministic function $\eta : S \times E \to S$. Different effects $e$ will change a state $s$ to a different next state $s' = \eta(s, e)$. The MDP transition model $T$ can be reconstructed from the CMDP by the equation:

$$T(s, a, s'; \tau) = P(s' \mid f(s), a) = \tau(s, a, e), \tag{1}$$

where $e$ is the effect of action $a$ that takes $s$ to $s'$, if such an $e$ exists, otherwise $T(s, a, s'; \tau) = 0$.

The benefit of this decomposition is that while there may be a large number of states, there is usually a limited number of definable effects of actions, and those are assumed to depend only on some features of the states and not on the actual states themselves. We can therefore turn the learning of the transition model into a supervised online classification problem that can be solved by any standard online classification method. More specifically, the classification task is to *predict* the effect $e$ of an action $a$ in a state $s$ with features $f(s)$.

### 2.2 A multi-view transfer framework

In our framework, the knowledge gathered and transferred by the agent is collected into a library $\mathcal{T}$ of online effect predictors or views.

A *view* consists of a structure component $\bar{f}$ that picks the features which should be focused on, and a quantitative component $\Theta$ that defines how these features should be combined to approximate the distribution of action effects. Formally, a *view* is defined as $\tau = (\bar{f}, \Theta)$, such that $P(E|S, a; \tau) = P(E|\bar{f}(S), a; \Theta) = \tau(S, a, E)$, in which $\bar{f}$ is an orthogonal projection of $f(s)$ to some subspace

of $R^d$. Each view $\tau$ is specialized in predicting the effects of one action $a(\tau) \in A$ and it yields a probability distribution for the effects of the action $a$ in any state. This prediction is based on the features of the state and the parameters $\Theta(\tau)$ of the view that may be adjusted based on the actual effects observed in the task environment.

We denote the subset of views that specify the effects for action $a$ by $\mathcal{T}^a \subset \mathcal{T}$. The main challenge is to build and maintain a comprehensive set of views that can be used in new environments likely resembling the old ones, but at the same time allow adaptation to new tasks with completely new transition dynamics and feature distributions.

At the beginning of every new task, the existing library is copied into a working library which is also augmented with fresh, uninformed views, one for each action, that are ready to be adapted to new tasks. We then select, for each action, a view with a good track record. This view is is used to estimate the optimal policy based on the transition model specified in Equation 1, and the policy is used to pick the first action $a$. The action effect is then used to score all the views in the working library and to adjust their parameters. In each round the selection of views is repeated based on their scores, and the new optimal policy is calculated based on the new selections. At the end of the task, the actual library is updated by possibly recruiting the views that have "performed well" and retiring those that have not. A more rigorous version of the procedure is described in Algorithm 1.

---

**Algorithm 1** TES: **T**ransferring **E**xpectation**s** using a library of views

---

**Input:** $\mathcal{T} = \{\tau_1, \tau_2, ...\}$: view library; CMDP$_j$: a new $j^{th}$ task; $\Phi$: view goodness evaluator
Let $\mathcal{T}_0$ be a set of fresh views - one for each action
$\mathcal{T}_{tmp} \leftarrow \mathcal{T} \cup \mathcal{T}_0$    /* THE WORKING LIBRARY FOR THE TASK */
**for all** $a \in A$ **do**   $\hat{T}[a] \leftarrow \text{argmax}_{\tau \in \mathcal{T}^a} \Phi(\tau, j)$   **end for**    /* SELECTING VIEWS */
**for** $t = 0, 1, 2, ...$ **do**
   $a_t \leftarrow \hat{\pi}(s_t)$, where $\hat{\pi}$ is obtained by solving MDP using transition model $\hat{T}$
   Perform action $a_t$ and observe effect $e_t$
   **for all** $\tau \in \mathcal{T}_{tmp}^{a_t} \cup \mathcal{T}^{a_t}$ **do**    $Score[\tau] \leftarrow Score[\tau] + \log \tau(s_t, a_t, e_t)$    **end for**
   **for all** $\tau \in \mathcal{T}_{tmp}^{a_t}$      **do**    Update view $\tau$ based on $(f(s_t), a_t, e_t)$    **end for**
   $\hat{T}[a_t] \leftarrow \text{argmax}_{\tau \in \mathcal{T}_{tmp}^{a_t}} Score[\tau]$    /* SELECTING VIEWS */
**end for**
**for all** $a \in A$ **do**    $\tau^* \leftarrow \text{argmax}_{\tau \in \mathcal{T}_{tmp}^a} Score[\tau]$;
                       $\mathcal{T}^a \leftarrow growLibrary(\mathcal{T}^a, \tau^*, Score, j)$    /* UPDATING LIBRARY */
**end for**
**if** $|\mathcal{T}| > M$ **then**    $\mathcal{T} \leftarrow \mathcal{T} - \{\text{argmin}_{\tau \in \mathcal{T}} \Phi(\tau, j)\}$    **end if**    /* PRUNING LIBRARY */

---

### 2.2.1 Scoring the views

To assess the quality of a view $\tau$, we measure its predictive performance by a cumulative log-score. This is a *proper* score [12] that can be effectively calculated online.

Given a sequence $D^a = (d_1, d_2, \ldots, d_N)$ of observations $d_i = (s_i, a, e_i)$ in which action $a$ has resulted in effect $e_i$ in state $s_i$, the score for an $a$-specialized $\tau$ is

$$S(\tau, D^a) = \sum_{i=1}^{N} \log \tau(s_i, a, e_i; \theta^{:i}(\tau)),$$

where $\tau(s_i, a, e_i; \theta^{:i}(\tau))$ is the probability of event $e_i$ given by the event predictor $\tau$ based on the features of state $s_i$ and the parameters $\theta^{:i}(\tau)$ that may have been adjusted using previous data $(d_1, d_2, \ldots, d_{i-1})$.

### 2.2.2 Growing the library

After completing a task, the highest scoring new views for each action are considered for recruiting into the actual library. The winning "newbies" are automatically accepted. In this case, the data has most probably come from the distribution that is far from the any current models, otherwise one of the current models would have had an advantage to adapt and win.

The winners $\tau^*$ that are adjusted versions of old views $\bar{\tau}$ are accepted as new members if they score significantly higher than their original versions, based on the logarithm of the prequential likelihood ratio [5] $\Lambda(\tau^*, \bar{\tau}) = S(\tau^*, D^a) - S(\bar{\tau}, D^a)$. Otherwise, the original versions $\bar{\tau}$ get their parameters updated to the new values. This procedure is just a heuristic and other inclusion and updating criteria may well be considered. The policy is detailed in Algorithm 2.

---

**Algorithm 2** Grow sub-library $\mathcal{T}^a$

---

**Input:** $\mathcal{T}^a, \tau^*, Score, j$: task index; $c$: constant; $H_{\tau^*} = \{\}$: empty history record
**Output:** updated library subset $\mathcal{T}^a$ and winning histories $H_{\tau^*}$
**case** $\tau^* \in \mathcal{T}_0^a$ **do** $\quad \mathcal{T}^a \leftarrow \mathcal{T}^a \cup \{\tau^*\}$ $\qquad\qquad$ /\* ADD NEWBIE TO LIBRARY \*/
**otherwise** $\qquad$ **do** $\quad$ Let $\bar{\tau} \in \mathcal{T}$ be the original, not adapted version of $\tau^*$
$\qquad\qquad\qquad\qquad$ **case** $Score[\tau^*] - Score[\bar{\tau}] > c$ **do** $\quad \mathcal{T}^a \leftarrow \mathcal{T}^a \cup \{\tau^*\}$
$\qquad\qquad\qquad\qquad$ **otherwise** $\qquad\qquad\qquad\qquad$ **do** $\quad \mathcal{T}^a \leftarrow \mathcal{T}^a \cup \{\tau^*\} - \{\bar{\tau}\}$
$\qquad\qquad\qquad\qquad\qquad\qquad\qquad\qquad\qquad$ $H_{\tau^*} \leftarrow H_{\bar{\tau}}$ $\quad$ /\* INHERIT HISTORY \*/
$H_{\tau^*} \leftarrow H_{\tau^*} \cup \{j\}$

---

### 2.2.3 Pruning the library

To keep the library relatively compact, a plausible policy is to remove views that have not performed well for a long time, possibly because there are better predictors or they have become obsolete in the new tasks or environments. To implement such a retiring scheme, each view $\tau$ maintains a list $H_\tau$ of task indices that indicates the tasks for which the view has been the best scoring predictor for its specialty action $a(\tau)$. We can then calculate the recency weighted track record for each view. In practice, we have adopted the procedure by Zhu et al. [27] that introduces the recency weighted score at time $T$ as

$$\Phi(\tau, T) = \sum_{t \in H_\tau} e^{-\mu(T-t)},$$

where $\mu$ controls the speed of decay of past success. Other decay functions could naturally also be used. The pruning can then be done by introducing a threshold for recency weighted score or always maintaining the top M views.

## 3 A view learning algorithm

In *TES*, a view can be implemented by any probabilistic classification model that can be quickly learned online. A popular choice for representing the transition model in factored domains is the dynamic Bayesian network (DBN), but learning DBNs is computationally very expensive. Recent studies [24, 25] have shown encouraging results in learning the structure of logistic regression models that can serve as local structures of DBNs. While these models cannot capture all the conditional distributions, their simplicity allows fast online learning in very high dimensional spaces.

We introduce an online sparse multinomial logistic regression algorithm to incrementally learn a view. The proposed algorithm is similar to so called group-lasso [26] which has been recently suggested for feature selection among a very large set of features [25].[1]

Assuming $K$ classes of vectors $x \in \mathbb{R}^d$, each class $k$ is represented with a $d$-dimensional prototype vector $W_k$. Classification of an input vector $x$ in logistic regression is based on how "similar" it is to the prototype vectors. Similarity is measured by the inner product $\langle W_k, x \rangle = \sum_{i=1}^d W_{ki}x_i$. The log probability of a class $y$ is defined by $\log P(y = k|x; W_k) \propto \langle W_k, x \rangle$. The classifier can then be parametrized by stacking the $W_k$ vectors as rows into a matrix $W = (W_1, ..., W_K)^T$.

An online learning system usually optimizes its probabilistic classification performance by minimizing a total loss function through updating its parameters over time. A typical item-wise loss function of a multinomial logistic regression classifier is $l(W) = -\log P(y|x; W)$, where $(y, x)$ denotes data item observed at time $t$. To achieve a parsimonious model in a feature-rich domain, we express our a priori belief that most features are superfluous by introducing a regularization term

$\Psi(W) = \lambda \sum_i^d \sqrt{K} ||W_{\cdot i}||_2$, where $||W_{\cdot i}||_2$ denotes the 2-norm of the $i^{th}$ column of $W$, and $\lambda$ is a positive constant. This regularization is similar to that of group lasso [26]. It communicates the idea that it is likely that a whole column of $W$ has zero values (especially, for large $\lambda$). A column of all zeros suggests that the corresponding feature is irrelevant for classification.

The objective function can now be written as $\sum_{t=1}^T l(W^t, d^t) + \Psi(W^t)$, where $W^t$ is the coefficient matrix learned using $t-1$ previously observed data items. Inspired by the efficient dual averaging method [24] for solving lasso and group lasso [25] logistic regression, we extend the results to the multinomial case. Specifically, the loss minimizing sequence of parameter matrices $W^t$ can be achieved by the following online update scheme.

Let $G_{ki}^t$ be the derivatives of function $l_t(W)$ with respect to $W_{ki}$. $\bar{G}^t$ is a matrix of average partial derivatives $\bar{G}_{ki}^t = \frac{1}{t} \sum_{j=1}^t G_{ki}^j$, where $G_{ki}^j = -x_i^j(I(y^j = k) - P(k|x^j; W^{j-1}))$.

Given a $K \times d$ average gradient matrix $\bar{G}^t$, and a regularization parameter $\lambda > 0$, the $i^{th}$ column of the new parameter matrix $W^{t+1}$ can be achieved as follows

$$W_{\cdot i}^{t+1} = \begin{cases} \vec{0} & \text{if } ||\bar{G}_{\cdot i}^t||_2 \leq \lambda\sqrt{K}, \\ \frac{\sqrt{t}}{\alpha}\left(\frac{\lambda\sqrt{K}}{||\bar{G}_{\cdot i}^t||_2} - 1\right)\bar{G}_{\cdot i}^t & \text{otherwise}, \end{cases} \tag{2}$$

where $\alpha > 0$ is a constant. The update rule (2) dictates that when the length of the average gradient matrix column is small enough, the corresponding parameter column should be truncated to zero. This introduces feature selection into the model.

## 4  Related work

The survey by Taylor and Stone [20] offers a comprehensive exposition of recent methods to transfer various forms of knowledge in RL. Not much research, however, has focused on transferring transition models. For example, while superficially similar to our framework, the case-based reasoning approaches [4] [13] focus on collecting good decisions instead of building models of world dynamics. Taylor proposes TIMBREL [19] to transfer observations in a source to a target task via manually tailored inter-task mapping. Fernandez et al. [7] transfers a library of policies learned in previous tasks to bias exploration in new tasks. The method assumes a constant inter-task state space, otherwise a state mapping strategy is needed.

Hester and Stone [8] describe a method to learn a decision tree for predicting state relative changes which are similar to our action effects. They learn decision trees online by repeatedly applying batch learning. Such a sequence of classifiers forms an effect predictor that could be used as a member of our view library. This work, however, does not directly focus on transfer learning.

Multiple models have previously been used to guide behavior in non-stationary environments [6] [15]. Unlike our work, these studies usually assume a common concrete state space. In representation selection, Konidaris and Barto [9] focus on selecting the best abstraction to assist the agent's skill learning, and Van et al. [21] study using multiple representations together to solve a RL problem. None of these studies, however, solve the problem of transferring knowledge in heterogeneous environments.

Atkeson and Santamaria introduce a locally weighted transfer learning technique called *LWT* to adapt previously learned transition models into a new situation [1]. This study is among the very few that actually consider transferring the transition model to a new task [20]. While their work is conducted in continuous state space using a fixed state similarity measure, it can be adapted to a discrete case. Doing so corresponds to adopting a fixed single view. We will compare our work with this approach in our experiments. This approach could also be extended to be compatible with our work by learning a library of state similarity measures and developing a method to choose among those similarities for each task.

Wilson et al. [23] also address the problem of transfer in heterogeneous environments. They formalize the problem as learning a generative Dirichlet process for MDPs and suggest an approximate solution using Gibbs sampling. Our method can be seen as a structure learning enhanced alternative implementation of this generative model. Our online-method is computationally more efficient, but the MCMC estimation should eventually yield more accurate estimates. Both models can also

be adjusted to deal with non-stationary task sources. The work by Wilson et al. demonstrates the method for reward models, and it is unclear how to extend the approach for transferring transition models. We will also compare our work with this hierarchical Bayes approach in our experiments.

# 5 Experiments

We examine the performance of our expectation transfer algorithm *TES* that transfers views to speed-up the learning process across different environments in two benchmark domains. We show that *TES* can efficiently: a) learn the appropriate views online, b) select views using the proposed scoring metric, c) achieve a good jump start, and d) perform well in the long run.

To better compare with some related work, we evaluate the performance of *TES* for transferring both transition models and reward models in RL. *TES* can be adapted to transfer reward models as follows: Assuming that the rewards follow a Gaussian distribution, a view of the expected reward model can be learned similarly as shown in section 3. We use an online sparse linear regression model instead of the multinomial logistic regression. Simply replacing matrix $W$ by a vector $w$, and using squared loss function, the coefficient update function can be found similar to that in Equation 2 [24]. When studying reward models, the transition models are assumed to be known.

## 5.1 Learning views for effective transfer

In the first experiment, we compare *TES* with the locally weighted *LWT* approach by Atkeson et al. [1] and the non-parametric hierarchical Bayesian approach *HB* by Wilson et al. [23] in transferring reward models. We adopt the same domain as described in Wilson et al.'s *HB* paper, but augment each state with 200 random binary features. The objective is to find the optimal route to a known goal state in a color maze. Assuming a deterministic transition model, the highest cumulative reward, determined by the colors around each cell/state, can be achieved on the optimal route.

*Experiment set-up*: Five different reward models are generated by normal Gaussian distributions, each depending on different sets of features. The start state is random. We run experiments on 15 tasks repeatedly 20 times, and conduct leave-one-task-out test. The maximum size $M$ of the views library, initially empty, is set to be 20; threshold $c$ for growing the library is set to be $\log 300$. The parameters for view learning are: $\lambda = 0.05$ and $\alpha = 2.5$.

Table 1: Transfer of reward models: Cumulative reward in the first episodes; Time to solve 15 tasks (in minutes), in which each is run with 200 episodes. Map sizes vary from $20 \times 20$ to $30 \times 30$.

| Methods | Tasks | | | | | | | | | | | | | | | Time |
|---|---|---|---|---|---|---|---|---|---|---|---|---|---|---|---|---|
| | 1 | 2 | 3 | 4 | 5 | 6 | 7 | 8 | 9 | 10 | 11 | 12 | 13 | 14 | 15 | |
| HB | -108.01 | -85.26 | -67.46 | -90.17 | -130.11 | **-95.42** | -46.23 | -77.10 | **-83.91** | **-51.01** | -131.44 | -97.05 | -90.11 | **-48.91** | -92.31 | 77.2 |
| LWT | -79.41 | -114.28 | -83.31 | **-46.70** | -245.11 | -156.23 | -47.05 | **-49.52** | -105.24 | -88.19 | -174.15 | -85.10 | **-55.45** | -101.24 | -86.01 | 28.6 |
| *TES* | **-45.01** | **-78.23** | **-62.15** | -54.46 | **-119.76** | -115.77 | **-37.15** | -58.09 | -167.13 | -59.11 | **-102.46** | **-45.99** | -86.12 | -67.23 | **-81.39** | 31.5 |

As seen in Table 1, *TES* on average wins over *HB* in 11 and *LWT* in 12 out of 15 tasks. In the $15 \times 20 = 300$ runs *TES* wins over *HB* 239 and over *LWT* 279 times, both yielding binomial test p-values less than $0.05$. This demonstrates that *TES* can successfully learn the views and utilize them in novel tasks. Moreover, *TES* runs much faster than *HB*, and just slightly slower than *LWT*. Since *HB* does not learn the relevant features for model representation, it may overfit, and the knowledge learned cannot be easily generalized. It also needs a costly sampling method. Similarly, the strategy for *LWT* that tries to learn one common model for transfer in various tasks often does not work well.

## 5.2 Multi-view transfer in complex environments

In the second experiment, we evaluate *TES* in a more challenging domain, transferring transition models. We consider a grid-based robot navigation problem in which each grid-cell has the surface of either sand, soil, water, brick, or fire. In addition, there may be walls between cells. The surfaces and walls determine the stochastic dynamics of the world. However, the agent also observes numerous other features in the environment. The agent has to learn to focus on the relevant features to quickly achieve its goal. The goal is to reach any exit door in the world consuming as little energy as possible.

*Experiment set-up*: The agent can perform four actions (move up, down, left, right) which will lead it to one of the four states around it, or leave it to its current state if it bumps into a wall. The agent will spend 0.01 units of energy to perform an action. It loses 1 unit if falling into a fire, but gains 1 unit when reaching an exit door. A task ends when the agent reaches any exit door or fire.

We design fifteen tasks with grid sizes ranging from $20 \times 20$ to $30 \times 30$. Each task has a different state space and different terminal states. Each state (cell) also has 200 irrelevant random binary features, besides its surface materials and the walls around it. The tasks may have different dynamics as well as different distributions of the surface materials. In our experiments, the environment transition dynamics is generated using three different sets of multinomial logistic regression models so that every combination of cell surfaces and walls around the cell will lead to a different transition dynamics at the cell. The probability of going through a wall is rounded to zero and the freed probability mass is evenly distributed to other effects. The agent's starting position is randomly picked in each episode.

We represent five effects of the actions: moved up, left, down, right, did not move. The maximum size $M$ of the view library, initially empty, is set to be 20; threshold $c = \log 300$. In a new environment, the *TES*-agent mainly relies on its transferred knowledge. However, we allow some $\epsilon$-greedy exploration with $\epsilon = 0.05$. The parameters for view learning algorithm are that $\lambda = 0.05$, $\alpha = 1.5$.

We conduct leave-one-out cross-validation experiment with fifteen different tasks. In each scenario the agent is first allowed to experience fourteen tasks, over 100 episodes in each, and it is then tested on the remaining one task. No recency weighting is used to calculate the goodness of the views in the library. We next discuss experimental results averaged over 20 runs showing 95% confidence intervals (when practical) for some representative tasks.

**Transferring expectations between homogeneous tasks.** To ensure that *TES* is capable of basic model transfer, we first evaluate it on a simple task to ensure that the learning algorithm in section 3 works. We train and test *TES* on two environments which have same dynamics and 200 irrelevant binary features that challenge agent's ability to learn a compact model for transfer. Figure 1a shows how much the other methods lose to *TES* in terms of accumulated reward in the test task. *loreRL* is an implementation of *TES* equipped with the view learning algorithm that does not transfer knowledge. *fRmax* is the factored *Rmax* [3] in which the network structures of transition models are provided by an oracle [17]; its parameter $m$ is set to be 10 in all the experiments. *fEpsG* is a heuristic in which the optimistic *Rmax* exploration of *fRmax* is replaced by an $\epsilon$-greedy strategy ($\epsilon = 0.1$). The results show that these oracle methods still have to spend time to learn the model parameters, so they gain less accumulated reward than *TES*. This also suggests that the transferred view of *TES* is likely not only compact but also accurate. Figure 1a further shows that *loreRL* and *fEpsG* are more effective than *fRmax* in early episodes.

**View selection vs. random views.** Figure 1b shows how different views lead to different policies and accumulated rewards over the first 50 episodes in a given task. The *Rands* curves show the accumulated reward difference to *TES* when the agent follows some random combinations of views from the library. For clarity we show only 5 such random combinations. For all these, the difference turns negative fast in the beginning indicating less reward in early episodes. We conclude that our view selection criterion outperforms random selection.

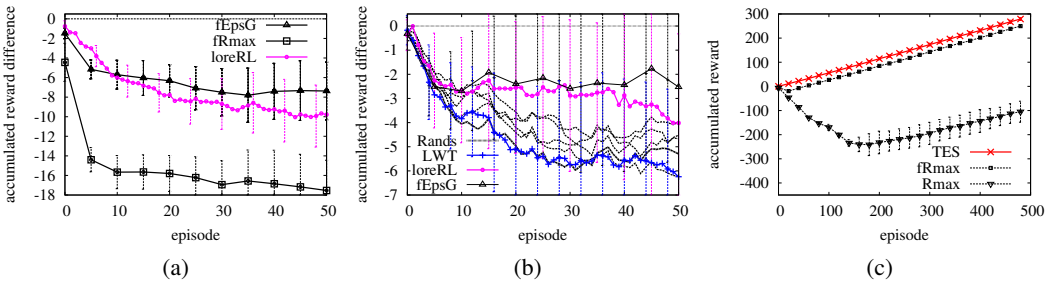

(a)                          (b)                          (c)

Figure 1: Performance difference to *TES* in early trials in a) homogeneous, b) heterogeneous environments. c) Convergence.

Table 2: Cumulative reward after first episodes. For example, in Task 1 *TES* can save $(0.616 - 0.113)/0.01 = 50.3$ actions compared to *LWT*.

| Methods | Tasks | | | | | | | | | | | | | | |
|---|---|---|---|---|---|---|---|---|---|---|---|---|---|---|---|
| | 1 | 2 | 3 | 4 | 5 | 6 | 7 | 8 | 9 | 10 | 11 | 12 | 13 | 14 | 15 |
| loreRL | -0.681 | -0.826 | -0.814 | -1.068 | -0.575 | -0.810 | -0.529 | -0.398 | -0.653 | -0.518 | -0.528 | -0.244 | -0.173 | -1.176 | -0.692 |
| LWT | 0.113 | -0.966 | -0.300 | **0.024** | -1.205 | **-0.345** | -1.104 | -1.98 | -0.057 | -0.664 | **-0.230** | -1.228 | 0.034 | 0.244 | -0.564 |
| *TES* | **0.616** | **-0.369** | **0.230** | -0.044 | **-0.541** | -0.784 | **-0.265** | **0.255** | **0.001** | **-0.298** | -1.184 | **-0.077** | **0.209** | **0.389** | **-0.407** |

**Multiple views vs. single view, and non-transfer.** We compare the multi-view learning *TES* agent with a non-transfer agent *loreRL*, and an *LWT* agent that tries to learn only one good model for transfer. We also compare with the oracle method *fEpsG*. As seen in Figure 1b, *TES* outperforms *LWT* which, due to differences in the tasks, also performs worse than *loreRL*. When the earlier training tasks are similar to the test task, the *LWT* agent performs well. However, the *TES* agent also quickly picks the correct views, thus we never lose much but often gain a lot. We also notice that *TES* achieves a higher accumulated reward than *loreRL* and *fEpsG* that are bound to make uninformed decisions in the beginning.

Table 2 shows the average cumulative reward after the first episode (the jumpstart effect) for each test task in the leave-one-out cross-validation. We observe that *TES* usually outperforms both the non-transfer and the *LWT* approach. In all $15 \times 20 = 300$ runs, *TES* wins over *LWT* 247 times and it wins over *loreRL* 263 times yielding p-values smaller than $0.05$.

We also notice that due to its fast capability of capturing the world dynamics, *TES* running time is just slightly longer than *LWT*'s and *loreRL*'s, which do not perform extra work for view switching but need more time and data to learn the dynamics models.

**Convergence.** To study the asymptotic performance of *TES*, we compare with the oracle method *fRmax* which is known to converge to a (near) optimal policy. Notice that in this feature-rich domain, *fRmax* without the pre-defined DBN structure is just similar to *Rmax*. Therefore, we also compare with *Rmax*. For *Rmax*, the number of visits to any state before it is considered "known" is set to $5$, and the exploration probability $\epsilon$ for known states starts to decrease from value $0.1$.

Figure 1c shows the accumulated rewards and their statistical dispersion over episodes. Average performance is reflected by the angles of the curves. As seen, *TES* can achieve a (near) optimal policy very fast and sustain its good performance over the long run. It is only gradually caught up by *fRmax* and *Rmax*. This suggests that *TES* can successfully learn a good library of views in heterogeneous environments and efficiently utilize those views in novel tasks.

# 6 Conclusions

We have presented a framework for learning and transferring multiple expectations or views about world dynamics in heterogeneous environments. When the environments are different, the combination of learning multiple views and dynamically selecting the most promising ones yields a system that can learn a good policy faster and gain higher accumulated reward as compared to the common strategy of learning just a single good model and using it in all occasions.

Utilizing and maintaining multiple models require additional computation and memory. We have shown that by a clever decomposition of the transition function, model selection and model updating can be accomplished efficiently using online algorithms. Our experiments demonstrate that performance improvements in multi-dimensional heterogeneous environments can be achieved with a small computational cost.

The current work addresses the question of learning good models, but the problem of learning good policies in large state spaces still remains. Our model learning method is independent of the policy learning task, thus it can well be coupled with any scalable approximate policy learning algorithms.

**Acknowledgments**

This research is supported by Academic Research Grants: MOE2010-T2-2-071 and T1 251RES1005 from the Ministry of Education in Singapore.

## Footnotes

[1]We report here the details of the method that should allow its replication. A more comprehensive description is available as a separate report in the supplementary material.

# References

[1] Atkeson, C., Santamaria, J.: A comparison of direct and model-based reinforcement learning. In: ICRA'97. vol. 4, pp. 3557–3564 (1997)

[2] Boutilier, C., Dearden, R., Goldszmidt, M.: Stochastic dynamic programming with factored representations. Journal of Artificial Intelligence 121, 49–107 (2000)

[3] Brafman, R.I., Tennenholtz, M.: R-max - a general polynomial time algorithm for near-optimal reinforcement learning. Journal of Machine Learning Research 3, 213–231 (2002)

[4] Celiberto, L.A., Matsuura, J.P., de Mntaras, R.L., Bianchi, R.A.C.: Using cases as heuristics in reinforcement learning: A transfer learning application. In: IJCAI'11. pp. 1211–1217 (2011)

[5] Dawid, A.: Statistical theory: The prequential approach. Journal of the Royal Statistical Society A 147, 278–292 (1984)

[6] Doya, K., Samejima, K., Katagiri, K.i., Kawato, M.: Multiple model-based reinforcement learning. Neural Computation 14, 1347–1369 (June 2002)

[7] Fernández, F., García, J., Veloso, M.: Probabilistic policy reuse for inter-task transfer learning. Robot and Autonomous System 58, 866–871 (July 2010)

[8] Hester, T., Stone, P.: Generalized model learning for reinforcement learning in factored domains. In: AAMAS'09. vol. 2, pp. 717–724 (2009)

[9] Konidaris, G., Barto, A.: Efficient skill learning using abstraction selection. In: IJCAI'09. pp. 1107–1112 (2009)

[10] Leffler, B.R., Littman, M.L., Edmunds, T.: Efficient reinforcement learning with relocatable action models. In: AAAI'07. vol. 1, pp. 572–577 (2007)

[11] McCarthy, J.: Situations, actions, and causal laws. Tech. Rep. Memo 2, Stanford Artificial Intelligence Project, Stanford University (1963)

[12] Savage, L.J.: Elicitation of personal probabilities and expectations. Journal of the American Statistical Association 66(336), 783–801 (1971)

[13] Sharma, M., Holmes, M., Santamaria, J., Irani, A., Isbell, C., Ram, A.: Transfer learning in real-time strategy games using hybrid cbr/rl. In: IJCAI'07. pp. 1041–1046 (2007)

[14] Sherstov, A.A., Stone, P.: Improving action selection in MDP's via knowledge transfer. In: AAAI'05. vol. 2, pp. 1024–1029 (2005)

[15] Silva, B.C.D., Basso, E.W., Bazzan, A.L.C., Engel, P.M.: Dealing with non-stationary environments using context detection. In: ICML'06. pp. 217–224 (2006)

[16] Soni, V., Singh, S.: Using homomorphisms to transfer options across continuous reinforcement learning domains. In: AAAI'06. pp. 494–499 (2006)

[17] Strehl, A.L., Diuk, C., Littman, M.L.: Efficient structure learning in factored-state MDPs. In: AAAI'07. pp. 645–650 (2007)

[18] Sutton, R.S., Barto, A.G.: Reinforcement Learning: An Introduction. MIT Press (1998)

[19] Taylor, M.E., Jong, N.K., Stone, P.: Transferring instances for model-based reinforcement learning. In: Machine Learning and Knowledge Discovery in Databases. LNAI, vol. 5212 (2008)

[20] Taylor, M.E., Stone, P.: Transfer learning for reinforcement learning domains: A survey. Journal of Machine Learning Research 10, 1633–1685 (December 2009)

[21] Van Seijen, H., Bakker, B., Kester, L.: Switching between different state representations in reinforcement learning. In: Proceedings of the 26th IASTED International Conference on Artificial Intelligence and Applications. pp. 226–231 (2008)

[22] Walsh, T.J., Li, L., Littman, M.L.: Transferring state abstractions between MDPs. In: ICML Workshop on Structural Knowledge Transfer for Machine Learning (2006)

[23] Wilson, A., Fern, A., Ray, S., Tadepalli, P.: Multi-task reinforcement learning: A hierarchical Bayesian approach. In: ICML'07. pp. 1015–1023 (2007)

[24] Xiao, L.: Dual averaging methods for regularized stochastic learning and online optimization. In: NIPS'09 (2009)

[25] Yang, H., Xu, Z., King, I., Lyu, M.R.: Online learning for group Lasso. In: ICML'10 (2010)

[26] Yuan, M., Lin, Y.: Model selection and estimation in regression with grouped variables. Journal of the Royal Statistical Society: Series B (Statistical Methodology) 68(1), 49–67 (2006)

[27] Zhu, X., Ghahramani, Z., Lafferty, J.: Time-sensitive Dirichlet process mixture models. Tech. Rep. CMU-CALD-05-104, School of Computer Science, Carnegie Mellon University (2005)

